# Dynamics of Attention as Near Saddle-Node Bifurcation Behavior

**Hiroyuki Nakahara***
General Systems Studies
University of Tokyo
3-8-1 Komaba, Meguro
Tokyo 153, Japan
nakahara@vermeer.c.u-tokyo.ac.jp

**Kenji Doya**
ATR Human Information Processing
Research Laboratories
2-2 Hikaridai, Seika, Soraku
Kyoto 619-02, Japan
doya@hip.atr.co.jp

## Abstract

In consideration of attention as a means for goal-directed behavior in non-stationary environments, we argue that the dynamics of attention should satisfy two opposing demands: long-term maintenance and quick transition. These two characteristics are contradictory within the linear domain. We propose the near saddle-node bifurcation behavior of a sigmoidal unit with self-connection as a candidate of dynamical mechanism that satisfies both of these demands. We further show in simulations of the 'bug-eat-food' tasks that the near saddle-node bifurcation behavior of recurrent networks can emerge as a functional property for survival in non-stationary environments.

## 1  INTRODUCTION

Most studies of attention have focused on the selection process of incoming sensory cues (Posner et al., 1980; Koch et al., 1985; Desimone et al., 1995). Emphasis was placed on the phenomena of causing different percepts for the same sensory stimuli. However, the selection of sensory input itself is not the final goal of attention. We consider attention as a means for goal-directed behavior and survival of the animal. In this view, dynamical properties of attention are crucial. While attention has to be maintained long enough to enable robust response to sensory input, it also has to be shifted quickly to a novel cue that is potentially important. Long-term maintenance and quick transition are critical requirements for attention dynamics.

We investigate a possible neural mechanism that enables those dynamical characteristics of attention.

First, we analyze the dynamics of a network of sigmoidal units with self-connections. We show that both long-term maintenance and quick transition can be achieved when the system parameters are near a "saddle-node bifurcation" point. Then, we test if such a dynamical mechanism can actually be helpful for an autonomously behaving agent in simulations of a 'bug-eat-food' task. The result indicates that near saddle-node bifurcation behavior can emerge in the course of evolution for survival in non-stationary environments.

## 2   NEAR SADDLE-NODE BIFURCATION BEHAVIOR

When a pulse-like input is given to a linear system, the rising and falling phases of the response have the same time constants. This means that long-term maintenance and quick transition cannot be simultaneously achieved by linear dynamics. Therefore, it is essential to consider a nonlinear dynamical mechanism to achieve these two demands.

### 2.1   DYNAMICS OF A SELF-RECURRENT UNIT

First, we consider the dynamics of a single sigmoidal unit with the self-connection weight $a$ and the bias $b$.

$$y(t+1) = F(ay(t)+b), \tag{1}$$

$$F(x) = \frac{1}{1+\exp(-x)}. \tag{2}$$

The parameters $(a, b)$ determine the qualitative behavior of the system such as the number of fixed points and their stabilities. As we change the parameters, the qualitative behavior of the system may suddenly change. This is referred to as "bifurcation" (Guckenheimer, et al., 1983). A typical example is a "saddle-node bifurcation" in which a pair of fixed points, one stable and one unstable, emerges. In our system, this occurs when the state transition curve $y(t+1) = F(ay(t)+b)$ is tangent to $y(t+1) = y(t)$. Let $y^*$ be this point of tangency. We have the following condition for saddle-node bifurcation.

$$F(ay^* + b) = y^* \tag{3}$$

$$\left.\frac{dF(ay+b)}{dy}\right|_{y=y^*} = 1 \tag{4}$$

These equations can be solved, by noting $F'(x) = F(x)(1 - F(x))$, as

$$a = \frac{1}{y^*(1-y^*)} \tag{5}$$

$$b = F^{-1}(y^*) - ay^* = F^{-1}(y^*) - \frac{1}{1-y^*} \tag{6}$$

By changing the fixed point value $y^*$ between 0 and 1, we can plot a curve in the parameter space $(a, b)$ on which saddle-node bifurcation occurs, as shown in Figure 1 (left). A pair of a saddle point and a stable fixed point emerges or disappears when the parameters pass across the cusp like curve (cases 2 and 4). The system has only one stable fixed point when the parameters are outside the cusp (case 1) and three fixed points inside the cusp (case 3).

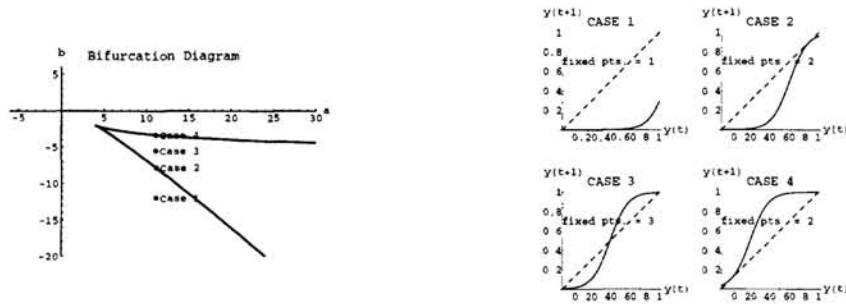

Figure 1: Bifurcation Diagram of a Self-Recurrent Unit. Left: the curve in the parameter space $(a, b)$ on which saddle-node bifurcation is seen. Right: state transition diagrams for four different cases.

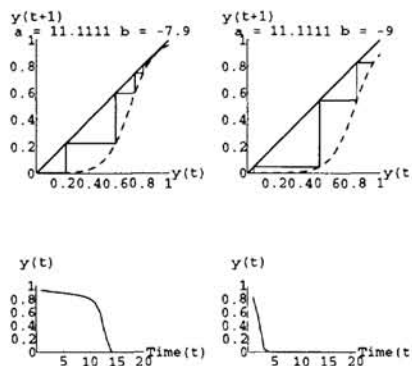

Figure 2: Temporal Responses of Self-Recurrent Units. Left: near saddle-node bifurcation. Right: far from bifurcation.

An interesting behavior can be seen when the parameters are just outside the cusp, as shown in Figure 2 (left). The system has only one fixed point near $y = 0$, but once the unit is activated ($y \simeq 1$), it stays "on" for many time steps and then goes back to the fixed point quickly. Such a mechanism may be useful in satisfying the requirements of attention dynamics: long-term maintenance and quick transition.

## 2.2 NETWORK OF SELF-RECURRENT UNITS

Next, we consider the dynamics of a network of the above self-recurrent units.

$$y_i(t+1) = F[ay_i(t) + b + \sum_{j, j \neq i} c_{ij} y_j(t) + d_i u_i(t)], \tag{7}$$

where $a$ is the self connection weight, $b$ is the bias, $c_{ij}$ is the cross connection weight, and $d_i$ is the input connection weight, and $u_i(t)$ is the external input. The effect of lateral and external inputs is equivalent to the change in the bias, which slides the sigmoid curve horizontally without changing the slope.

For example, one parameter set of the bifurcation at $y^* = 0.9$ is $a = 11.11$ and $b \simeq -7.80$. Let $b = -7.90$ so that the unit has a near saddle-node bifurcation behavior when there is no lateral or external inputs. For a fixed $a = 11.11$, as we increase $b$, the qualitative behavior of the system appears as case 3 in Figure 1, and

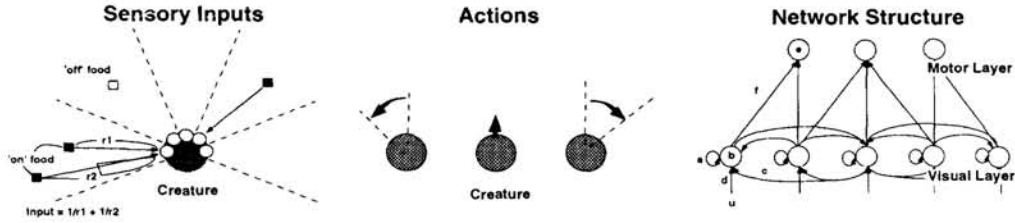

Figure 3: A Creature's Sensory Inputs(Left), Motor System(Center) and Network Architecture(Right)

then, it changes again at $b \simeq -3.31$, where the fixed point at $y = 0.1$, or another bifurcation point, appears as case 4 in Figure 1. Therefore, if the input sum is large enough, i.e. $\sum_{j,j \neq i} c_{ij} y_j + d_i u_j > -3.31 - (-7.90) \simeq 4.59$, the lower fixed point at $y = 0.1$ disappears and the state jumps up to the upper fixed point near $y = 1$, quickly turning the unit "on". If the lateral connections are set properly, this can in turn suppress the activation of other units. Once the external input goes away, as we see in Figure 2 (left), the state stays "on" for a long time until it returns to the fixed point near $y = 0$.

# 3 EVOLUTION OF NEAR BIFURCATION DYNAMICS

In the above section, we have theoretically shown the potential usefulness of near saddle-node bifurcation behavior for satisfying demands for attention dynamics. We further hypothesize that such behavior is indeed useful in animal behaviors and can be found in the course of learning and evolution of the neural system.

To test our hypothesis, we simulated a 'bug-eat-food' task. Our purpose in this simulation was to see whether the attention dynamics discussed in the previous section would help obtain better performance in a non-stationary environment. We used evolutionary programming (Fogel et al, 1990) to optimize the performance of recurrent networks and feedforward networks.

## 3.1 THE BUG AND THE WORLD

In our simulation, a simple creature traveled around a non-stationary environment. In the world, there were a certain number of food items. Each item was fixed at a certain place in the world but appeared or disappeared in a stochastic fashion, as determined by a two-state Markov system. In order to survive, A creature looked for food by traveling the world. The amount of food a creature found in a certain time period was the measure of its performance.

A creature had five sensory inputs, each of which detected food in the sector of 45 degrees (Figure 3, right). Its output level was given by $\sum_j \frac{1}{r_j}$, where $r_j$ was the distance to the $j$-th food item within the sector. Note that the format of the input contained information about distance and also that the creature could only receive the amount of the input but could not distinguish each food from others.

For the sake of simplicity, we assumed that the creature lived in a grid-like world. On each time step, it took one of three motor commands: L: turn left (45 degrees),

| Density of Food | 0.05 | | 0.10 | |
|---|---|---|---|---|
| Markov Transition Matrix | .5 .5 | .8 .8 | .5 .5 | .8 .8 |
| of each food | .5 .5 | .2 .2 | .5 .5 | .2 .2 |
| Random Walk | 7.0 | 6.9 | 13.8 | 13.9 |
| Nearest Visible | 42.7 | 18.6 | 65.3 | 32.4 |
| FeedForward | 58.6 | 37.3 | 84.8 | 60.0 |
| Recurrent | 65.7 | 43.6 | 94.0 | 66.1 |
| Nearest Visible/Invisible | 97.7 | 97.1 | 129.1 | 128.8 |

Table 1: Performances of the Recurrent Network and Other Strategies.

C: step forward, and R: turn right (Figure 3, center). Simulations were run with different Markov transition matrices of food appearance and with different food densities. A creature got the food when it reached the food, whether it was visible or invisible. When a creature ate a food item , a new food item was placed randomly. The size of the world was 10x10 and both ends were connected as a torus.

A creature was composed of two layers: visual layer and motor layer (Figure 3, left). There were five units[1] in visual layer, one for each sensory input, and their dynamics were given by Equation (7). The self-connection $a$, the bias $b$ and the input weight $d_i$ were the same for all units. There were three units in motor layer, each coding one of three motor commands, and their state was given by

$$x_k(t) = e_k + \sum_i f_{ki} y_i(t), \tag{8}$$

$$p_k(t) = \frac{\exp(x_k(t))}{\sum_l \exp(x_l(t))}, \tag{9}$$

where $e_k$ was the bias and $f_{ki}$ was the feedforward connection weight.[2] One of the three motor commands (L,C,R) was chosen stochastically with the probability $p_k$ ($k$=L,C,R). The activation pattern in visual layer was shifted when the creature made a turn, which should give proper mapping between the sensory input and the working memory.

## 3.2  EVOLUTIONARY PROGRAMMING

Each recurrent network was characterized by the parameters $(a, b, c_{ij}, d_i, e_k, f_{ki})$, some of which were symmetrically shared, e.g. $c_{12} = c_{21}$. For comparison, we also tested feedforward networks where recurrent connections were removed, i.e. $a = c_{ij} = 0$.

A population of 60 creatures was tested on each generation. The initial population was generated with random parameters. Each of the top twenty scoring creatures produced three offspring; one identical copy of the parameters of the parent's and two copies of these parameters with a Gaussian fluctuation. In this paper, we report the result after 60 generations.

## 3.3  PERFORMANCE

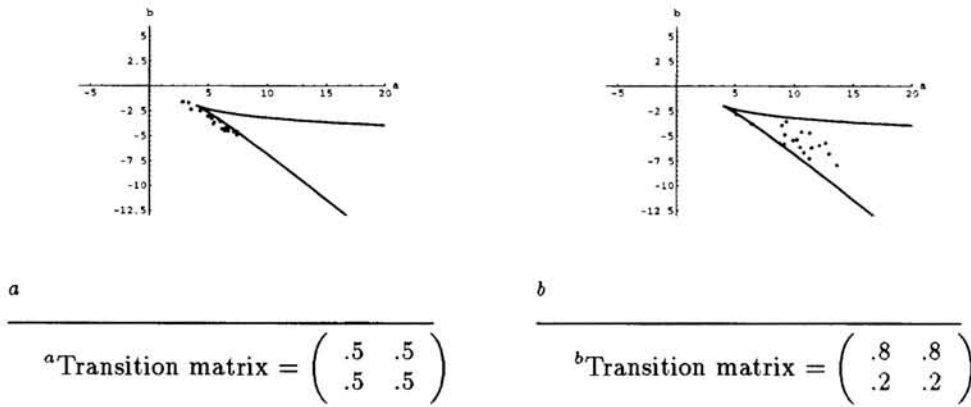

$$^a\text{Transition matrix} = \begin{pmatrix} .5 & .5 \\ .5 & .5 \end{pmatrix} \qquad ^b\text{Transition matrix} = \begin{pmatrix} .8 & .8 \\ .2 & .2 \end{pmatrix}$$

Figure 4: The Convergence of the Parameter of $(a, b)$ by Evolutionary Programming Plotted in the Bifurcation Diagram. The food density is 0.10 in both examples above.

Table 1 shows the average of food found after 60 generations. As a reference of performance level, we also measured the performances of three other simple algorithms: 1) random walk: one of the three motor commands is taken randomly with equal probability. 2) nearest visible: move toward the nearest food visible at the time within the creature's field of view of $(u_2, u_3, u_4)$. 3) nearest visible/invisible: move toward the nearest food within the view of $(u_2, u_3, u_4)$ no matter if it is visible or not, which gives an upper bound of performance.

The performance of recurrent network is better than that of feedforward network and 'nearest visible'. This suggests that the ability of recurrent network to remember the past is advantageous.

The performance of feedforward network is better than that of 'nearest visible'. One reason is that feedforward network could cover a broader area to receive inputs than 'nearest visible'. In addition, two factors, the average time in which a creature reaches the food and the average time in which the food disappears, may influence the performance of feedforward network and 'nearest visible'. Feedforward network could optimize its output to adapt two factors with its broader view in evolution while 'nearest visible' did not have such adaptability.

It should be noted that both of 'nearest visible/invisible' and 'nearest visible' explicitly assumed the higher-order sensory processing: distinguishing each food item from the others and measuring the distance between each food and its body. Since its performance is so different regardless of its higher-order sensory processing, it implies the importance of remembering the past. We can regard recurrent network as compromising two characteristics, remembering the past as 'nearest visible/invisible' did and optimizing the sensitivity as feedforward network did, although recurrent network did not have a perfect memory as 'nearest visible/invisible'.

## 3.4   CONVERGENCE TO NEAR-BIFURCATION REGIME

We plotted the histogram of the performance in each generation and the history of the performance of a top-scoring creature over generations. Though they are not shown here, the performance was almost optimal after 60 generations.

Figure 4 shows that two examples of a graph in which we plotted the parameter

set $(a, b)$ of top twenty scoring creatures in the 60th generation in the bifurcation diagram. In the left graph, we can see the parameter set has converged to a regime that gives a near saddle-node bifurcation behavior. On the other hand, in the right graph, the parameter set has converged into the inside of cusp. It is interesting to note that the area inside of the cusp gives bistable dynamics. Hence, if the input is higher than a repelling point, it goes up and if the input is lower, it goes down. The reason of the convergence to that area is because of the difference of the world setting, that is, a Markov transition matrix. Since food would disappear more quickly and stay invisible longer in the setting of the right graph, it should be beneficial for a creature to remember the direction of higher inputs longer. In most of cases reported in Table 1, we obtained the convergence into our predicted regime and/or the inside of the cusp.

## 4  DISCUSSION

Near saddle-node bifurcation behavior can have the long-term maintenance and quick transition, which characterize attention dynamics. A recurrent network has better performance than memoryless systems for tasks in our simulated non-stationary environment. Clearly, near saddle-node bifurcation behavior helped a creature's survival and in fact, creatures actually evolved to our expected parameter regime. However, we also obtained the convergence into another unexpected regime which gives bistable dynamics. How the bistable dynamics are used remains to be investigated.

### Acknowledgments

H.N. is grateful to Ed Hutchins for his generous support, to John Batali and David Fogel for their advice on the implementation of evolutionary programming and to David Rogers for his comments on the manuscript of this paper.

## Footnotes

*currently at Dept. of Cognitive Science and Institute for Neural Computation, U. C. San Diego, La Jolla CA 92093-0515. hnakahar@cogsci.ucsd.edu

[1]We denote each unit in visual layer by $u_1, u_2, u_3, u_4, u_5$ from the left to the right for the later convenience

[2]In this simulation reported here, we set $e_k = 0$.

### References

R. Desimone, E. K. Miller, L. Chelazzi, & A. Lueschow. (1995) Multiple Memory Systems in the Visual Cortex. In M. Gazzaniga (ed.), *The Cognitive Neurosciences*, 475-486. MIT Press.

D. B. Fogel, L. J. Fogel, & V. W. Porto. (1990) Evolving Neural Networks. *Biological cybernetics* **63**:487-493.

J. Guckenheimer & P. Homes. (1983) *Nonlinear Oscillations, Dynamical Systems, and Bifurcation of Vector Fields*

C. Koch & S. Ullman. (1985) Shifts in selective visual attention:towards the underlying neural circuitry. *Human Neurobiology* **4**:219-227.

M. Posner, C. .R .R. Snyder, & B. J. Davidson. (1980) Attention and the detection of signals. *Journal of Experimental Psychology: General* **109**:160-174